# Neuronal Adaptation for Sampling-Based Probabilistic Inference in Perceptual Bistability

**David P. Reichert, Peggy Seriès, and Amos J. Storkey**
School of Informatics, University of Edinburgh
10 Crichton Street, Edinburgh, EH8 9AB
{d.p.reichert@sms., pseries@inf., a.storkey@} ed.ac.uk

## Abstract

It has been argued that perceptual multistability reflects probabilistic inference performed by the brain when sensory input is ambiguous. Alternatively, more traditional explanations of multistability refer to low-level mechanisms such as neuronal adaptation. We employ a Deep Boltzmann Machine (DBM) model of cortical processing to demonstrate that these two different approaches can be combined in the same framework. Based on recent developments in machine learning, we show how neuronal adaptation can be understood as a mechanism that improves probabilistic, sampling-based inference. Using the ambiguous Necker cube image, we analyze the perceptual switching exhibited by the model. We also examine the influence of spatial attention, and explore how binocular rivalry can be modeled with the same approach. Our work joins earlier studies in demonstrating how the principles underlying DBMs relate to cortical processing, and offers novel perspectives on the neural implementation of approximate probabilistic inference in the brain.

## 1 Introduction

Bayesian accounts of cortical processing posit that the brain implements a probabilistic model to learn and reason about the causes underlying sensory inputs. The nature of the potential cortical model and its means of implementation are hotly debated. Of particular interest in this context is bistable perception, where the percept switches over time between two interpretations in the case of an ambiguous stimulus such as the Necker cube, or two different images that are presented to either eye in binocular rivalry [1]. In these cases, ambiguous or conflicting sensory input could result in a bimodal posterior over image interpretations in a probabilistic model, and perceptual bistability could reflect the specific way the brain explores and represents this posterior [2, 3, 4, 5, 6]. Unlike more classic explanations that explain bistability with low-level mechanism such as neuronal fatigue (e.g. [7, 8]), maybe making it more of an epiphenomenon, the probabilistic approaches see bistability as a fundamental aspect of how the brain implements probabilistic inference.

Recently, it has been suggested that the cortex could employ *approximate* inference schemes, e.g. by estimating probability distributions with a set of samples, and studies show how electrophysiological [9] and psychophysical [10] data can be interpreted in that light. Gershman et al. [6] focus on binocular rivalry and point out how in particular Markov Chain Monte Carlo (MCMC) algorithms, where correlated samples are drawn over time to approximate distributions, might naturally account for aspects of perceptual bistability, such as its stochasticity and the fact that perception at any point in time only reflects an individual interpretation of the image rather than a full distribution over possibilities. Gershman et al. do not provide a concrete neural model, however.

In earlier work, we considered Deep Boltzmann Machines (DBMs) as models of cortical perception, and related hierarchical inference in these generative models to hallucinations [11] and attention

[12]. With the connection between MCMC and bistability established, it is natural to explore DBMs as models of bistability as well, because Gibbs sampling, a MCMC method, can be performed to do inference. Importantly from a neuroscientific perspective, Gibbs sampling in Boltzmann machines simply corresponds to the 'standard' way of running the DBM as a neural network with stochastic firing of the units. However, it is well known that MCMC methods in general and Gibbs sampling in particular can be problematic in practice for complex, multi-modal distributions, as the sampling algorithm can get stuck in individual modes ('the chain does not *mix*'). In very recent machine learning work, Breuleux at al. [13] introduced a heuristic algorithm called Rates Fast Persistent Contrastive Divergence (rates-FPCD) that aims to improve sampling performance in a Boltzmann machine model by dynamically changing the model parameters, such as the connection strengths. In closely related work, Welling [14] suggested a potential connection to dynamic synapses in the brain. Hence, neuronal adaptation, here meant to be temporary changes to neuronal excitability and synaptic efficacy, could actually be seen as a means of enhancing sampling based inference [2].

We thus aim to demonstrate how the low-level and probabilistic accounts of bistable perception can be combined. We present a biological interpretation of rates-FPCD in terms of neuronal adaptation, or neuronal fatigue and synaptic depression specifically. Using a DBM that was trained on the two interpretations of the Necker cube, we show how such adaptation leads to bistable switching of the internal representations when the model is presented with the actual ambiguous Necker cube. Moreover, we model the role of spatial attention in biasing the perceptual switching. Finally, we explore how the same approach can be applied also to binocular rivalry.

## 2   Neuronal adaptation in a Deep Boltzmann Machine

In this section we briefly introduce the DBM, the rates-FPCD algorithm as it is was motivated from a machine learning perspective, and then explain the latter's relation to biology.

A DBM [15] consists of stochastic binary units arranged hierarchically in several layers, with symmetric connections between layers and no connections within a layer. The first layer contains the visible units that are clamped to data, such as images, during inference, whereas the higher layers contain hidden units that learn representations from which they can generate the data in the visibles. With the states in layer $k$ denoted by $\mathbf{x}^{(k)}$, connection weights $\mathbf{W}^{(k)}$ and biases $\mathbf{b}^{(k)}$, the probability for a unit to switch on is determined by the input it gets from adjacent layers, using a sigmoid activation function:

$$P(x_i^{(k)} = 1 | \mathbf{x}^{(k-1)}, \mathbf{x}^{(k+1)}) = \left( 1 + \exp\left( -\sum_l w_{li}^{(k-1)} x_l^{(k-1)} - \sum_m w_{im}^{(k)} x_m^{(k+1)} - b_i^{(k)} \right) \right)^{-1}. \quad (1)$$

Running the network by switching units on and off in this manner implements Gibbs sampling on a probability distribution determined by an energy function $E$,

$$P(\mathbf{x}) \propto \exp(-E(\mathbf{x})) \quad \text{with} \quad E(\mathbf{x}) = \sum_k -\mathbf{x}^{(k)T} \mathbf{W}^{(k)} \mathbf{x}^{(k+1)} - \mathbf{x}^{(k)T} \mathbf{b}^{(k)}. \quad (2)$$

Intuitively speaking, when run the model performs a random walk in the energy landscape shaped during learning, where it is attracted to ravines. Jumping between high-probability modes of the distribution corresponds to traversing from one ravine to another.

### 2.1   Rates-FPCD, neuronal fatigue and synaptic depression

Unfortunately, for many realistically complex inference tasks MCMC methods such as Gibbs are prone to get stuck in individual modes, resulting in an incomplete exploration of the distribution, and there is much work in machine learning on improving sampling methods. One recently introduced algorithm is rates-FPCD (Rates Fast Persistent Contrastive Divergence) [13], which was utilized to sample from Restricted Boltzmann Machines (RBMs), the two layer building blocks of DBMs. Rates-FPCD is based on FPCD [16], which is used for training. Briefly, in FPCD one contribution to the weight training updates requires the model to be run continuously and independently of the data to explore the probability distribution as it is currently learned. Here it is important that the model does not get stuck in individual modes. It was found that introducing a fast changing component to

the weights (and biases) to dynamically and temporarily change the energy landscape can alleviate this problem. These fast weights $\mathbf{W}_f$, which are added to the actual weights $\mathbf{W}$, and the analogue fast biases $\mathbf{b}_f^{(k)}$ are updated according to

$$\mathbf{W}_f \quad \leftarrow \quad \alpha\mathbf{W}_f + \epsilon(\mathbf{x}^{(0)}p(\mathbf{x}^{(1)}|\mathbf{x}^0) - \mathbf{x}'^{(0)}\mathbf{x}'^{(1)T}), \tag{3}$$

$$\mathbf{b}_f^{(0)} \quad \leftarrow \quad \alpha\mathbf{b}_f^{(0)} + \epsilon(\mathbf{x}^{(0)} - \mathbf{x}'^{(0)}), \tag{4}$$

$$\mathbf{b}_f^{(1)} \quad \leftarrow \quad \alpha\mathbf{b}_f^{(1)} + \epsilon(p(\mathbf{x}^{(1)}|\mathbf{x}^0) - \mathbf{x}'^{(1)}). \tag{5}$$

Here, the visibles $\mathbf{x}^{(0)}$ are clamped to the current data item.[1] $\mathbf{x}'^{(0)}$ and $\mathbf{x}'^{(1)}$ are current samples from the freely run model. $\epsilon$ is a parameter determining the rate of adaptation, and $\alpha \leq 1$ is a decay parameter that limits the amount of weight change contributed by the fast weights. The second term in each of the parentheses has the effect of changing the weights and biases such that whatever states are currently being sampled by the model are made less likely in the following. Hence, this will eventually 'push' the model out of a mode it is stuck in. The first terms in the parentheses are computed over the data and leads to the model being drawn to states supported by the current input.

Computation of the first terms in the parentheses in equations 3-5 requires the training data. To turn FPCD into a general sampling algorithm applicable outside of training, when the training data is no longer around, rates-FPCD simply replaces the first terms with the so-called *rates*, which are the pairwise and unitary statistics averaged over *all* training data:

$$\mathbf{W}_f \quad \leftarrow \quad \alpha\mathbf{W}_f + \epsilon(E[\mathbf{x}^{(0)}\mathbf{x}^{(1)T}] - \mathbf{x}'^{(0)}\mathbf{x}'^{(1)T}), \tag{6}$$

$$\mathbf{b}_f^{(0)} \quad \leftarrow \quad \alpha\mathbf{b}_f^{(0)} + \epsilon(E[\mathbf{x}^{(0)}] - \mathbf{x}'^{(0)}), \tag{7}$$

$$\mathbf{b}_f^{(1)} \quad \leftarrow \quad \alpha\mathbf{b}_f^{(1)} + \epsilon(E[\mathbf{x}^{(1)}] - \mathbf{x}'^{(1)}) \tag{8}$$

($\mathbf{x}^{(1)}$ is sampled conditioned on the data). The rates are to be computed during training, but can then be used for sampling afterwards. It was found that these terms sufficiently serve to stabilize the sampling scheme, and that rates-FPCD yielded improved performance over Gibbs sampling [13].

Let us consider equations 6-8 from a biological perspective, interpreting the weight parameters as synaptic strengths and the biases as some overall excitability level of a neuron. The equations suggest that the capability of the network to explore the state space is improved by dynamically adjusting the neuron's parameters (cf. e.g. [17]) depending on the current states of the neuron and its connected partners (second terms in parentheses), drawing them towards some set values (first terms, the rate statistics). All that is needed for the latter is that the neuron stores its average firing activity during learning (for the bias statistics) and the synapses remember some average firing correlation between connected neurons (for the weight statistics). In particular, if activation patterns in the network are sparse and neurons are off most of the time, then these average terms will be rather low. During inference,[2] the neuron will fire strongly for its preferred stimulus (or stimulus interpretation), but then its firing probability will decrease as its excitability and synaptic efficacy drop, allowing the network to discover potential alternative interpretations of the stimulus. Thus, in the case of sparse activity, equations 6-8 implement a form of neuronal fatigue and synaptic depression.

Preceding the introduction of rates-FPCD as a sampling algorithm, we also utilized the same mechanism (but only applied to the biases) in a biological model of hallucinations [11] to model homeostatic [18] regulation of neuronal firing. We showed how it helps to make the system more robust against noise corruption in the input, though it can lead to hallucinations under total sensory deprivation. Hence, the same underlying mechanisms could either be understood as short-term neuronal adaptation or longer term homeostatic regulation, depending on the time scales involved.

## 3 Experiments: Necker cube

We trained a DBM on binary images of cubes at various locations, representing the two unambiguous interpretations of the Necker cube, and then tested the model on the actual, ambiguous Necker cube

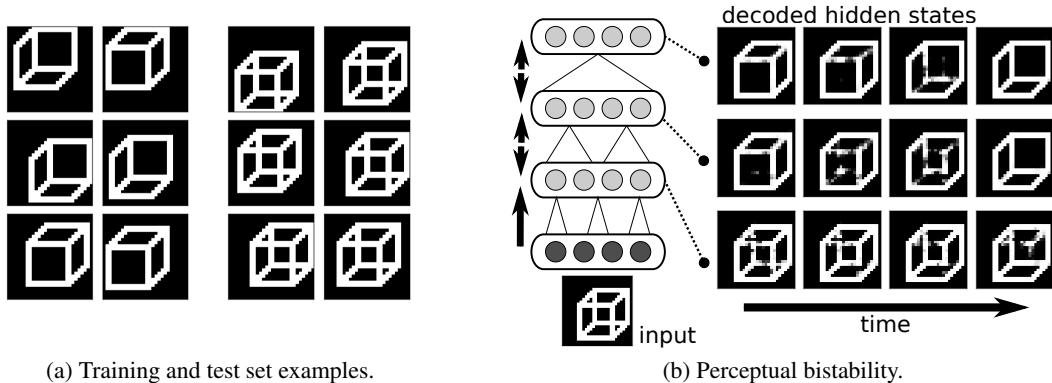

(a) Training and test set examples.　　　　　　　(b) Perceptual bistability.

Figure 1: *(a)*: Examples of the unambiguous training images (left) and the ambiguous test images (right). *(b)*: During inference on an ambiguous image, the decoded hidden states reveal perceptual switching resulting from neuronal adaptation. Four consecutive sampling cycles are shown.

(Figure 1a). We use a similar setup[3] to that described in [11, 12], with localized receptive fields the size of which increased from lower to higher hidden layers, and sparsity encouraged simply by initializing the biases to negative values in training. As in the aforementioned studies, we are interested in what is inferred in the hidden layers as the image is presented in the visibles, and 'decode' the hidden states by computing a reconstructed image for each hidden layer. To this end, starting with the states of the hidden layer of interest, the activations (i.e. firing probabilities) in each subsequent lower layer are computed deterministically in a single top-down pass, doubling the weights to compensate for the lack of bottom-up input, until a reconstructed image is obtained in the visibles. In this way, the reconstructed image is determined by the states in the initial layer alone, independently of the actual current states in the other layers.

When presented with a Necker cube image, the hidden states were found to converge within a few sampling cycles (each consisting of one up and one down pass of sampling all hidden layers) to one of the unambiguous interpretations and remained therein, exhibiting no perceptual switching to the respective alternative interpretation.[4] We then employed rates-FPCD to model neuronal adaptation.[5] It should be noted that unlike in [13], we utilize it in a DBM rather than a RBM, and during inference instead of when generating data samples (i.e. in our case the visibles are always clamped to an image). The rate statistics were computed by measuring unit activities and pairwise correlations when the trained model was run on the training data. With neuronal adaption, the internal representations as decoded from the hidden layer were found to switch over time between the two image interpretations, thus the model exhibited perceptual bistability.

An example of the switching of internal representations is displayed in Figure 1b. It can be observed that the perceptual state is most distinct in higher layers. For quantitative analysis, we computed the squared reconstruction error of the image decoded from the topmost layer with regards to either of the two image interpretations. Plotted against time (Figure 2a), this shows how the internal representations evolve during a trial. The representations match one of the two image interpretations in a relatively stable manner over several sampling cycles, with some degradation before and a short transition phase during a perceptual switch.

To examine the effects of adaptation on an individual neuron, we picked a unit in the top layer that showed high variance in both its activity levels and neuronal parameters as they changed over the

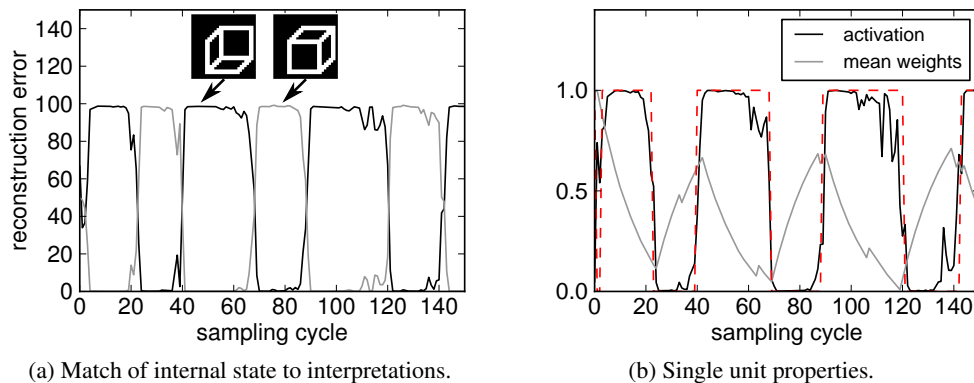

(a) Match of internal state to interpretations.  (b) Single unit properties.

Figure 2: *(a)*: Time course of squared reconstruction errors of the decoded topmost hidden states w.r.t. either of the two image interpretations. Apart from during the transition periods, the percept at any point matches one (close to zero error) but not the other interpretation (high error). *(b)*: Activation (i.e. firing probability) and mean synaptic strength (arbitrary origin and units) of a top layer unit that participates in coding for one but not the other interpretation (dashed line marks currently active interpretation). Depression and recovery of synaptic efficacy during instantiation of the preferred and non-preferred interpretations, respectively, lead to changes in activation that precede the next perceptual switch.

trial, indicating that this unit was involved in coding for one but not the other image interpretation. In Figure 2b are plotted the time course of its activity levels (i.e. firing probability according to equation 1) and the mean synaptic efficacy, i.e. weight strength, of connections to this unit.[6] As expected, the firing probability of this unit is close to one for one of the interpretations and close to zero for the other, especially in the initial time period after a perceptual switch. However, as the neuron's firing rate and synaptic activity deviate from their low average levels, the synaptic efficacy changes as shown in the plot. For example, during instantiation of the preferred stimulus interpretation, the drop of neuronal excitability ultimately leads to a waning of activity that precedes and, together with the changes in the overall network, subsequently triggers the next perceptual switch.

For another trial where we used an image of the Necker cube in a different position, the same unit showed constant low firing rates, indicating that it was not involved in representing that image. The neuronal parameters were then found to be stable throughout the trial, after a slight initial monotonic change that would allow the neuron to assume its low baseline activity as determined by the rate statistics. Moreover, other units were found to have relatively stable high firing rate for a given image throughout the trial, coding for features of the stimulus that were common to both image interpretations, even though their neuronal parameters equally adapted due to their elevated activity. This is due to the extent of adaptation being limited by the decay parameter $\alpha$ (equations 6-8), and shows that the adaptation can be set to be sufficiently strong to allow for exploration of the posterior, without overwhelming the representations of unambiguous image features. Similarly, we note that internal representations of the model when presented with the unambiguous images from the training set were stable under adaptation with our setting of parameter values.

We also quantified the statistics of perceptual switching by measuring the length of time the model's state would stay in either of the two interpretations for one of the test images. The resulting histograms of percept durations, i.e. time intervals between switches, are displayed in Figure 3a separately for the two interpretations. They are shaped like gamma or log-normal distributions, qualitatively in agreement with experimental results in human subjects [19]. There is a bias apparent in the model towards one of the interpretations (different for different images). Some biases are observed in humans (as visible in the data in [4]), potentially induced by statistical properties of the environment. However, our data set did not involve any biases, so this seems to be merely an artifact produced by the (basic) training procedure used.

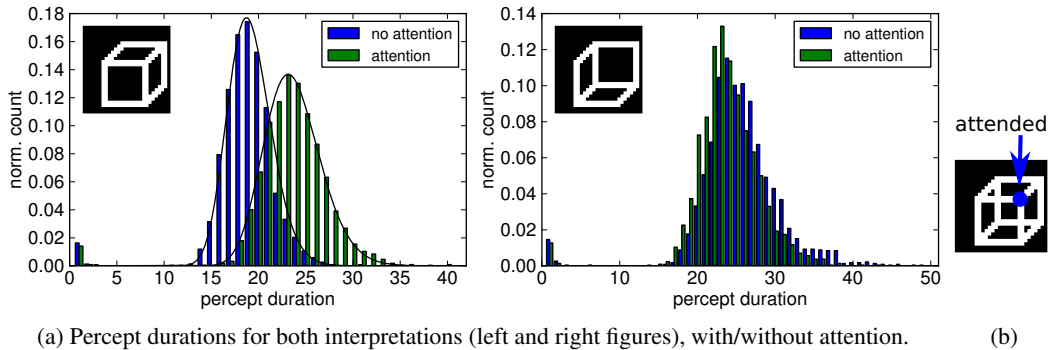

(a) Percept durations for both interpretations (left and right figures), with/without attention. (b)

Figure 3: *(a)*: Histograms over percept durations between perceptual switches, for either interpretation (*left* and *right*, respectively) of one of the test images. Ignoring the peaks at small interval lengths, which stem from fluctuations during transitions, the histograms are very well fitted by log-normal distributions (black curves, omitted in right figure to avoid clutter). Also plotted in both figures are histograms with spatial attention employed (see Section 3.1) to one of the interior corners of the Necker cube (as shown in *(b)*). The distributions shift or remain unchanged depending on whether the attended corner is salient or not for the image interpretation in question.

## 3.1 The role of spatial attention

The statistics of multistable perception can be influenced voluntarily by human subjects [20]. For the Necker cube, overtly directing one's gaze to corners of the cube, especially the interior ones, can have a biasing effect [21]. This could be explained by these features being in some way more salient for either of the two interpretations. An explanation matching our (simplified) setup would be that opaque cubes (as used in training) uniquely match one of the interpretations and lack one of the two interior corners. In the following, we model not eye movements but covert attention, involving only the shifting of an internal attentional 'spotlight', which also has been shown to affect perceptual switching in the Necker cube [22].[7]

The presented image remained unchanged and a spatial spotlight that biased the internal representations of the model was employed in the first hidden layer. To implement the spotlight, we made use of the fact that receptive fields were topographically organized, and that sparsity in a DBM breaks the symmetry between units being on and off and makes it possible to suppress represented information by suppressing the activity of specific hidden units [12]. We used a Gaussian shaped spotlight that was centered at one of the salient internal corners of the Necker cube (Figure 3b) and applied it to the hidden units as additional negative biases, attenuating activity further away from the focus.

The effect of attention on the percept durations for one of the test images are displayed in Figure 3a, together with the data obtained without attention for comparison. For the interpretation that matched the corner that was attended, we found a shift towards longer percept durations (Figure 3a, left), whereas the distribution for the other interpretation was relatively unchanged (Figure 3a, right). Averaged over all test images, the mean interval spent representing the interpretation favored by spatial attention saw a 25% increase vs. approx. no change for the other interpretation. Hence, in the model spatial attention prolongs the percept whose salient feature is being attended. This seems to be qualitatively in line with experimental data at least in terms of voluntary attention having an effect, although specifics can depend on the nature of the stimulus and the details of the instructions given to experimental subjects [23].

## 4 Experiments: binocular rivalry

Several related studies that considered perceptual multistability in the light of probabilistic inference focused on binocular rivalry [2, 5, 6]. There, human observers are presented with a different image to each eye, and their perception is found to switch between the two images. Depending on

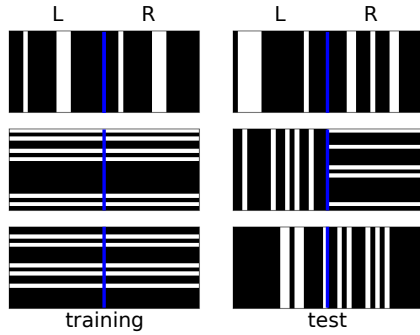

L    R       L    R

training         test

Figure 4: : Example images for the binocular rivalry experiment. Training images (*left*) contained either horizontal or vertical bars, and the left and right image halves were identical (corresponding to the left and right 'eyes'). For the test images (*right*), the left and right halves are drawn independently. They could come from the same category (top and bottom examples) or from conflicting categories (middle example).

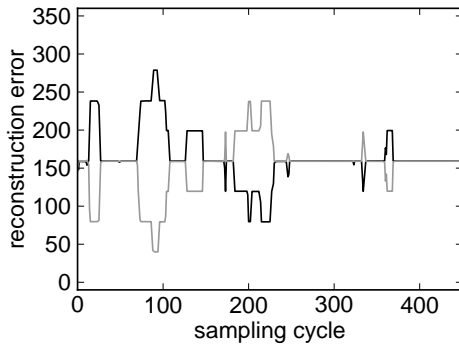

(a) Percept vs. eye images for same category.

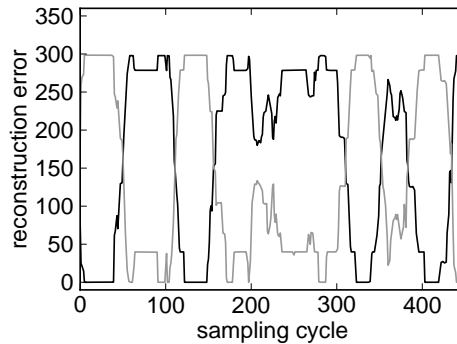

(b) Percept vs. eye images for conflict. categories.

Figure 5: For binocular rivalry, displayed are the squared reconstruction errors for decoded top layer representations computed against either of the two input images. *(a)*: The input images came from the same category (here, vertical bars), and fusing of the percept was prominent, resulting in modest, similar errors for both images. *(b)*: For input images from conflicting categories, the percept alternated more strongly between the images, although intermediate, fused states were still more prevalent than was the case for the Necker cube. The step-like changes in the error were found to result from individual bars appearing and disappearing in the percept.

specifics such as size and content of the images, perception can switch completely between the two images, fuse them, or do either to varying degrees over time [24, 25]. We demonstrate with a simple experiment that the phenomenon of binocular rivalry can be addressed in our framework as well.

To this end, the same model architecture as before was used, but the number of visible units was doubled and the units were separated into left and right 'eyes'. During training, both sets of visibles simply received the same images. During testing however, the left and right halves were set to independently drawn training images to simulate the binocular rivalry experiment. The units in the first hidden layer were set to be monocular in the sense that their receptive fields covered visible units only in either of the left or right half, whereas higher layers did not made this distinction. As a data set we used images containing either vertical or horizontal bars (Figure 4).

As with the Necker cube, perceptual switching was observed with adaptation but not without. Generally, the perceptual state was found to be biased to one of the two images for some periods, while fusing the images to some extent during transition phases (Figure 5). Interestingly, whether fusing or alternation was more prominent depended on the nature of the conflict in the two input images: For images from the same category (both vertical *or* horizontal lines), fusing occurred more often (Figure 5a), whereas for images from conflicting categories, the percept represented more distinctly either image and fusing happened primarily in transition periods (Figure 5b). We quantified this by computing the reconstruction errors from the decoded hidden states with regards to the two images, and taking the absolute difference averaged over the trial as measure for how much the internal states were representing both images individually rather than fused versions. We found that this measure was more than two times higher for conflicting categories. This result is qualitatively in line with psychophysical experiments that showed fusing for differing but compatible images (e.g. different patches of the same source image) [24, 25].

# 5 Related work and discussion

Our study contributes to the emerging trend in computational neuroscience to consider approximate probabilistic inference in the brain (e.g. [9, 10]), and complements several recent papers that examine perceptual multistability in this light. Gershman et al. [6] argued for interpreting multistability as inference based on MCMC, focusing on binocular rivalry only. Importantly, they use Markov random fields as a high-level description of the perceptual problem itself (two possible 'causes' generating the image, with a topology matching the stimulus). They argue that the brain might implement MCMC inference over these external variables, but do not make any statement w.r.t. the underlying neural mechanisms. In contrast, in our model MCMC is performed over the *internal*, neurally embodied latent variables that were learned from data. Bistability results from bimodality in the learned high-dimensional hidden representations, rather than directly from the problem formulation. In another study, Sundareswara and Schrater [4] model perceptual switching for the Necker cube, including the influence of image context, which we could explore in future work. Similar to [6], they start from a high-level description of the problem. They design a custom abstract inference process that makes different predictions from our model: In their model, samples are drawn i.i.d. from the two posterior modes representing the two interpretations and are accumulated over time, with older samples being exponentially discounted. A separate decision process selects from the samples and determines what interpretation reaches awareness. In our model, the current conscious percept is simply determined by the current overall state of the network, and the switching dynamics are a direct result of how this state evolves over time (as in [6]).

Hohwy et al. [5] explain binocular rivalry descriptively in their predictive coding framework. They identify switching with exploration in an energy landscape, and suggest the contribution of stochasticity or adaptation, but they do not make the connection to sampling and do not provide a computational model. The work by Grossberg and Swaminathan [8] is an example of a non-probabilistic model of, among other things, Necker cube bistability, providing much biological detail, and considering the role of spatial attention. Their study is also an instance of an approach that bases the switching on neuronal adaptation, but does not see a functional role for multistability as such, relegating instead the functional relevance of adaptation to a role it plays during learning only. Similarly, in earlier work Dayan [2] utilizes an ad-hoc adaptation process in a deterministic probabilistic model of binocular rivalry. He suggests sampling could provide stochasticity, wondering about the relation between sampling and adaptation. This is was what we have addressed here. Indeed, our approach is supported by recent psychophysics results [26], which indicate that *both* noise and neuronal adaptation are necessary to explain binocular rivalry.

We note that our setup is of course a simplification and abstraction in that we do not explicitly model depth. Indeed, in perceiving the Necker cube one does not see the actually opaque cubes we used in training, but rather a 3D wireframe cube. Peculiarly, this is actually contrary to the depth information available, as a (2D) image of a cube is not actually a 3D cube, but collection of lines on a flat surface. How is a paradoxically 'flat 3D cube' represented in the brain? In a hierarchical architecture consisting of specialized areas, this might be realized by having a high level area that codes for objects (e.g. area IT in the cortex) represent a 3D cube, whereas another area that is primarily involved with depth as such represents a flat surface. Our work here and earlier [11, 12] showed that in a DBM, different hidden layers can represent different and partially conflicting information (cf. Figure 1b). Finally, we also note that in preliminary experiments with depth information (using real valued visibles) perceptual switching did still occur.

In conclusion, we provided a biological interpretation of rates-FPCD, and thus showed how two seemingly distinct explanations for perceptual multistability, probabilistic inference and neuronal adaptation, can be merged in one framework. Unlike other approaches, our account combines sampling based inference and adaptation in a concrete neural architecture utilizing learned representations of images. Moreover, our study further demonstrates the relevance of DBMs as cortical models [11, 12]. We believe that further developing hybrid approaches – combining probabilistic models, dynamical systems, and classic connectionist networks – will help identifying the neural substrate of the Bayesian brain hypothesis.

**Acknowledgments**

Supported by the EPSRC, MRC and BBSRC. We thank N. Heess and the reviewers for comments.

## Footnotes

[1] In practice, minibatches are used.

[2] Applied in a DBM, not a RBM; see next section.

[3]Images of 28x28 pixels, three hidden layers with 26x26 units each. Pretraining of the layers with CD-1, no training of full DBM.

[4]It should be noted that the behavior of the network will depend heavily on the specifics of the training and the data set used. We employed only the most simple training methods – layer-wise pre-training with CD-1 and no tuning of the full DBM – and do not claim that more advanced methods could not lead to better sampling behavior, especially for this simple toy data. Indeed, using PCD instead we found some spontaneous switching, though reconstructions were noisy. But for the argument at hand it is more important that *in general*, bad mixing with these models can be a problem that might be alleviated by methods such as rates-FPCD, hence using a setup that exhibits this problem is useful to make the point.

[5]$\alpha = 0.95$, $\epsilon = 0.001$ for Necker cube, $\alpha = 0.9$, $\epsilon = 0.002$ for binocular rivalry (Section 4).

[6]The changes to weights and biases are equivalent, so we show only the former.

[7]We did not find an experimental study examining covert attention on the interior corners in unmodified Necker cubes, which is what we simulate.

# References

[1] Leopold and Logothetis (1999) Multistable phenomena: changing views in perception. *Trends in Cognitive Sciences*, **3**, 254–264, PMID: 10377540.

[2] Dayan, P. (1998) A hierarchical model of binocular rivalry. *Neural Computation*, **10**, 1119–1135.

[3] van Ee, R., Adams, W. J., and Mamassian, P. (2003) Bayesian modeling of cue interaction: bistability in stereoscopic slant perception. *Journal of the Optical Society of America A*, **20**, 1398–1406.

[4] Sundareswara, R. and Schrater, P. R. (2008) Perceptual multistability predicted by search model for Bayesian decisions. *Journal of Vision*, **8**, 1–19.

[5] Hohwy, J., Roepstorff, A., and Friston, K. (2008) Predictive coding explains binocular rivalry: An epistemological review. *Cognition*, **108**, 687–701.

[6] Gershman, S., Vul, E., and Tenenbaum, J. (2009) Perceptual multistability as markov chain monte carlo inference. *Advances in Neural Information Processing Systems 22*.

[7] Blake, R. (1989) A neural theory of binocular rivalry. *Psychological Review*, **96**, 145–167, PMID: 2648445.

[8] Grossberg, S. and Swaminathan, G. (2004) A laminar cortical model for 3D perception of slanted and curved surfaces and of 2D images: development, attention, and bistability. *Vision Research*, **44**, 1147–1187.

[9] Fiser, J., Berkes, B., Orban, G., and Lengyel, M. (2010) Statistically optimal perception and learning: from behavior to neural representations. *Trends in Cognitive Sciences*, **14**, 119–130.

[10] Vul, E., Goodman, N. D., Griffiths, T. L., and Tenenbaum, J. B. (2009) One and done? optimal decisions from very few samples. *Proceedings of the 31st Annual Conference of the Cognitive Science Society.*.

[11] Reichert, D. P., Seriès, P., and Storkey, A. J. (2010) Hallucinations in Charles Bonnet Syndrome induced by homeostasis: a Deep Boltzmann Machine model. *Advances in Neural Information Processing Systems 23*, **23**, 2020–2028.

[12] Reichert, D. P., Seriès, P., and Storkey, A. J. (2011) A hierarchical generative model of recurrent Object-Based attention in the visual cortex. *Proceedings of the International Conference on Artificial Neural Networks (ICANN-11)*.

[13] Breuleux, O., Bengio, Y., and Vincent, P. (2011) Quickly generating representative samples from an RBM-Derived process. *Neural Computation*, pp. 1–16.

[14] Welling, M. (2009) Herding dynamical weights to learn. *Proceedings of the 26th Annual International Conference on Machine Learning*, Montreal, Quebec, Canada, pp. 1121–1128, ACM.

[15] Salakhutdinov, R. and Hinton, G. (2009) Deep Boltzmann machines. *Proceedings of the 12th International Conference on Artificial Intelligence and Statistics (AISTATS)*, vol. 5, pp. 448–455.

[16] Tieleman, T. and Hinton, G. (2009) Using fast weights to improve persistent contrastive divergence. *Proceedings of the 26th Annual International Conference on Machine Learning*, Montreal, Quebec, Canada, pp. 1033–1040, ACM.

[17] Maass, W. and Zador, A. M. (1999) Dynamic stochastic synapses as computational units. *Neural Computation*, **11**, 903–917.

[18] Turrigiano, G. G. (2008) The self-tuning neuron: synaptic scaling of excitatory synapses. *Cell*, **135**, 422–435, PMID: 18984155.

[19] Zhou, Y. H., Gao, J. B., White, K. D., Yao, K., and Merk, I. (2004) Perceptual dominance time distributions in multistable visual perception. *Biological Cybernetics*, **90**, 256–263.

[20] Meng, M. and Tong, F. (2004) Can attention selectively bias bistable perception? differences between binocular rivalry and ambiguous figures. *Journal of Vision*, **4**.

[21] Toppino, T. C. (2003) Reversible-figure perception: mechanisms of intentional control. *Perception & Psychophysics*, **65**, 1285–1295, PMID: 14710962.

[22] Peterson, M. A. and Gibson, B. S. (1991) Directing spatial attention within an object: Altering the functional equivalence of shape descriptions. *Journal of Experimental Psychology: Human Perception and Performance*, **17**, 170–182.

[23] van Ee, R., Noest, A. J., Brascamp, J. W., and van den Berg, A. V. (2006) Attentional control over either of the two competing percepts of ambiguous stimuli revealed by a two-parameter analysis: means do not make the difference. *Vision Research*, **46**, 3129–3141, PMID: 16650452.

[24] Tong, F., Meng, M., and Blake, R. (2006) Neural bases of binocular rivalry. *Trends in Cognitive Sciences*, **10**, 502–511.

[25] Knapen, T., Kanai, R., Brascamp, J., van Boxtel, J., and van Ee, R. (2007) Distance in feature space determines exclusivity in visual rivalry. *Vision Research*, **47**, 3269–3275, PMID: 17950397.

[26] Kang, M. and Blake, R. (2010) What causes alternations in dominance during binocular rivalry? *Attention, Perception, & Psychophysics*, **72**, 179–186.

